# String Kernels, Fisher Kernels and Finite State Automata

**Craig Saunders**     **John Shawe-Taylor**     **Alexei Vinokourov**
Department of Computer Science
Royal Holloway, University of London
Email: {craig, jst, alexei}@cs.rhul.ac.uk

## Abstract

In this paper we show how the generation of documents can be thought of as a $k$-stage Markov process, which leads to a Fisher kernel from which the $n$-gram and string kernels can be re-constructed. The Fisher kernel view gives a more flexible insight into the string kernel and suggests how it can be parametrised in a way that reflects the statistics of the training corpus. Furthermore, the probabilistic modelling approach suggests extending the Markov process to consider sub-sequences of varying length, rather than the standard fixed-length approach used in the string kernel. We give a procedure for determining which sub-sequences are informative features and hence generate a Finite State Machine model, which can again be used to obtain a Fisher kernel. By adjusting the parametrisation we can also influence the weighting received by the features. In this way we are able to obtain a logarithmic weighting in a Fisher kernel. Finally, experiments are reported comparing the different kernels using the standard Bag of Words kernel as a baseline.

## 1   Introduction

Recently the string kernel [6] has been shown to achieve good performance on text-categorisation tasks. The string kernel projects documents into a feature space indexed by all $k$-tuples of symbols for some fixed $k$. The strength of the feature indexed by the $k$-tuple $u = (u_1, \ldots, u_k)$ for a document $d$ is the sum over all occurrences of $u$ as a subsequence (not necessarily contiguous) in $d$, where each occurrence is weighted by an exponentially decaying function of its length in $d$. This naturally extends the idea of an $n$-gram feature space where the only occurrences considered are contiguous ones.

The dimension of the feature space and the non-sparsity of even modestly sized documents makes a direct computation of the feature vector for the string kernel infeasible. There is, however, a dynamic programming recursion that enables the semi-efficient evaluation of the kernel [6]. String kernels are apparently making no use of the semantic prior knowledge that the structure of words can give and yet they have been used with considerable success.

The aim of this paper is to place the $n$-gram and string kernels in the context of probabilistic modelling of sequences, showing that they can be viewed as Fisher kernels of a Markov generation process. This immediately suggests ways of introducing weightings derived from refining the model based on the training corpus.

Furthermore, this view also suggests extending consideration to subsequences of varying lengths in the same model. This leads to a Finite State Automaton again inferred from the data. The refined probabilistic model that this affords gives rise to two Fisher kernels depending on the parametrisation that is chosen, if we take the Fisher information matrix to be the identity.

We give experimental evidence suggesting that the new kernels are capturing useful properties of the data while overcoming the computational difficulties of the original string kernel.

## 2 The Fisher view of the $n$-gram and String kernels

In this section we show how the string kernel can be thought of as a type of Fisher kernel [2] where the fixed-length subsequences used as the features in the string kernel correspond to the parameters for building the model. In order to give some insight into the kernel we first give a Fisher formulation of the $n$-gram kernel (i.e. the string kernel which considers only *contiguous* sequences), and then extend this to the full string kernel.

Let us assume that we have some document $d$ of length $s$ which is a sequence of symbols belonging to some alphabet $A$, i.e. $d_i \in A$, $i = 1, \ldots, s$. We can consider document $d$ as being generated by a $k$-stage Markov process. According to this view, for sequences $u \in A^{k-1}$ we can define the probability of observing a symbol $x$ after a sequence $u$ as $p_{u \to x}$. Sequences of $k$ symbols therefore index the parameters of our model. The probability of a document $d$ being generated by the model is therefore

$$\mathbf{P}(d) = \prod_{j=k}^{|d|} p_{d[j-k+1:j-1] \to d_j},$$

where we use the notation $d[i : j]$ to denote the sequence $d_i d_{i+1} \cdots d_j$. Now taking the derivative of the log-probability:

$$\frac{\partial \ln \mathbf{P}(d)}{\partial p_{u \to x}} = \frac{\partial \ln \prod_{j=k}^{|d|} p_{d[j-k+1:j-1] \to d_j}}{\partial p_{u \to x}}$$

$$= \sum_{j=k}^{|d|} \frac{\partial \ln p_{d[j-k+1:j-1] \to d_j}}{\partial p_{u \to x}} = \frac{\mathrm{tf}(ux, d)}{p_{u \to x}} \qquad (1)$$

where $\mathrm{tf}(ux, d)$ is the term frequency of $ux$ in $d$, that is the number of times the string $ux$ occurs in $d$.[1]

_______________

[1] Since the $p_{u \to x}$ are not independent it is not possible to take the partial derivative of one parameter without affecting others. However we can approximate our approach:

We introduce an extra character $c$. For each $(n-1)$-gram $u$ we assign a sufficiently small probability to $p_{u \to c}$ and change the other $p_{u \to x}$ to $\hat{p}_{u \to x} = p_{u \to x}(1 - p_{u \to c})$. We now replace each occurence of $p_{u \to c}$ in $\mathbf{P}(d)$ by $1 - \sum_{a \in A \backslash \{c\}} \hat{p}_{u \to a}$. Thus, since $uc$ never occurs in $d$ and $\hat{p}_{u \to x} \approx p_{v \to x}$, the $u \to x$ Fisher score entry for a document $d$ becomes

$$\frac{\partial \mathbf{P}(d)}{\partial \hat{p}_{u \to x}} = \sum_{j=k}^{|d|} \frac{\partial \ln \hat{p}_{d[j-k+1:j-1] \to d_j}}{\partial \hat{p}_{u \to x}} = \frac{\mathrm{tf}(ux, d)}{\hat{p}_{u \to x}} - \frac{\mathrm{tf}(uc, d)}{p_{u \to c}} \approx \frac{\mathrm{tf}(ux, d)}{p_{u \to x}}.$$

The Fisher kernel is subsequently defined to be

$$k(d, d') = U_d^T I^{-1} U_{d'},$$

where $U_d$ is the Fisher score vector with $ux$-component $\frac{\partial \ln \mathbf{P}(d)}{\partial p_{u \to x}}$ and $I = \mathbf{E}_d[U_d U_d^T]$. It has become traditional to set the matrix $I$ to be the identity when defining a Fisher kernel, though this undermines the very satisfying property of the pure definition that it is independent of the parametrisation. We will follow this same route mainly to reduce the complexity of the computation. We will, however, subsequently consider alternative parameterisations.

Different choices of the parameters $p_{u \to x}$ give rise to different models and hence different kernels. It is perhaps surprising that the $n$-gram kernel is recovered (up to a constant factor) if we set $p_{u \to x} = |A|^{-1}$ for all $u \in A^{n-1}$ and $x \in A$, that is the least informative parameter setting. This follows since the feature vector of a document $d$ has entries

$$\phi_v(d) = \text{tf}(ux, d)/|A|^{-1}.$$

We therefore recover the $n$-gram kernel as the Fisher kernel of a model which uses a uniform distribution for generating documents.

Before considering how the $p_{u \to x}$ might be chosen non-uniformly we turn our attention briefly to the string kernel.

We have shown that we can view the $n$-gram kernel as a Fisher kernel. A little more work is needed in order to place the full string kernel (which considers non-contiguous subsequences) in the same framework.

First we define an index set $S_{k-1,q}$ over all (possibly non-contiguous) subsequences of length $k$, which finish in position $q$,

$$S_{k-1,q} = \{\mathbf{i} : 1 \leq i_1 < i_2 < \ldots < i_{k-1} < i_k = q\}.$$

We now define a probability distribution $\mathbf{P}_{S_{k-1,q}}$ over $S_{k-1,q}$ by weighting sequence $\mathbf{i}$ by $\lambda^{l(\mathbf{i})}$, where $l(\mathbf{i}) = i_k - i_1 + 1$ is the length of $\mathbf{i}$, and normalising with a fixed constant $C$. This may leave some probability unaccounted for, which can be assigned to generating a spurious symbol. We denote by $d[\mathbf{i}]$ the sequence of characters $d_{i_1} d_{i_2} \ldots d_{i_k}$. We now define a text generation model that generates the symbol for position $q$ by first selecting a sequence $\mathbf{i}$ from $S_{k-1,q}$ according to the fixed distribution $\mathbf{P}_{S_{k-1,q}}$ and then generates the next symbol based on $p_{d[\mathbf{i}'] \to d_{i_k}}$ for all possible values of $d_q$ where $\mathbf{i}' = (i_1, i_2, \ldots, i_{k-1})$ is the vector $\mathbf{i}$ without its last component. We will refer to this model as the *Generalised k-stage Markov model with decay factor* $\lambda$. Hence, if we assume that distributions are uniform

$$
\begin{aligned}
\frac{\partial \ln \mathbf{P}(d)}{\partial p_{u \to x}} &= \frac{\partial \ln \prod_{j=k}^{|d|} \sum_{\mathbf{i} \in S_{k-1,j}} \mathbf{P}_{S_{k-1,j}}(\mathbf{i}) p_{d[\mathbf{i}'] \to d_{i_k}}}{\partial p_{u \to x}} \\
&= \sum_{j=k}^{|d|} \frac{\partial \ln \sum_{\mathbf{i} \in S_{k-1,j}} \mathbf{P}_{S_{k-1,j}}(\mathbf{i}) p_{d[\mathbf{i}'] \to d_{i_k}}}{\partial p_{u \to x}} \\
&= |A| \sum_{j=k}^{|d|} \sum_{\mathbf{i} \in S_{k-1,j}} \mathbf{P}_{S_{k-1,j}}(\mathbf{i}) \chi_{ux}(d[\mathbf{i}]) \\
&= |A| C^{-1} \sum_{j=k}^{|d|} \sum_{\mathbf{i} \in S_{k-1,j}} \lambda^{l(\mathbf{i})} \chi_{ux}(d[\mathbf{i}]),
\end{aligned}
$$

where $\chi_{ux}$ is the indicator function for string $ux$. It follows that the corresponding Fisher features will be the weighted sum over all subsequences with decay factor $\lambda$. In other words we recover the string kernel.

**Proposition 1** *The Fisher kernel of the generalised $k$-stage Markov model with decay factor $\lambda$ and constant $p_{u \to x}$ is the string kernel of length $k$ and decay factor $\lambda$.*

## 3    The Finite State Machine Model

Viewing the $n$-gram and string kernels as Fisher kernels of Markov models means we can view the different sequences of $k - 1$ symbols as defining states with the next symbol controlling the transition to the next state. We therefore arrive at a finite state automaton with states indexed by $A^{k-1}$ and transitions labelled by the elements of $A$. Hence, if $u \in A^{k-1}$ the symbol $x \in A$ causes the transition to state $v[2:k]$, where $v = ux$.

One drawback of the string kernel is that the value of $k$ has to be chosen a-priori and is then fixed. A more flexible approach would be to consider different length subsequences as features, depending on their frequency. Subsequences that occur very frequently should be given a low weighting, as they do not contain much information in the same way that stop words are often removed from the bag of words representation. Rather than downweight such sequences an alternative strategy is to extend their length. Hence, the 3-gram `com` could be very frequent and hence not a useful discriminator. By extending it either backwards or forwards we would arrive at subsequences that are less frequent and so potentially carry useful information. Clearly, extending a sequence will always reduce its frequency since the extension could have been made in many distinct ways all of which contribute to the frequency of the root $n$-gram.

As this derivation follows more naturally from the analysis of the $n$-gram kernel described in Section 2 we will only consider contiguous subsequences also known as substrings. We begin by introducing the general Finite State Machine (FSM) model and the corresponding Fisher kernel.

**Definition 2** *A Finite State Machine model over an alphabet $A$ is a triple $\mathcal{F} = (\Sigma, \delta, p)$ where*

1.  *the non-empty set $\Sigma$ of states is a finite subset of $A^\star = \bigcup_{i=0}^{\infty} A^i$ that is closed under taking substrings,*

2.  *the transition function $\delta$*
    $$\delta : \Sigma \times A \longrightarrow \Sigma,$$
    *is defined by*
    $$\delta(u, x) = v[j : l(v)], \quad where \ v = ux \ and \ j = \min\{j : v[j : l(v)] \in \Sigma\},$$
    *if the minimum is defined, otherwise the empty sequence $\epsilon$*

3.  *for each state $u$ the function $p$ gives a function $p_u$, which is either a distribution over next symbols $p_u(x)$ or the all one function $p_u(x) = 1$, for $u \in \Sigma$ and $x \in A$.*

Given an FSM model $\mathcal{F} = (\Sigma, \delta, p)$ to process a document $d$ we start at the state corresponding to the empty sequence $\epsilon$ (guaranteed to be in $\Sigma$ as it is non-empty and closed under taking substrings) and follow the transitions dictated by the symbols

of the document. The probability of a document in the model is the product of the values on all of the transitions used:

$$\mathbf{P}_{\mathcal{F}}(d) = \prod_{j=1}^{|d|} p_{d[i_j:j-1]}(d_j),$$

where $i_j = \min\{i : d[i : j-1] \in \Sigma\}$. Note that requiring that the set $\Sigma$ to be closed under taking substrings ensures that the minimum in the definition of $\delta$ is always defined and that $d[i_j : j]$ does indeed define the state at stage $j$ (this follows from a simple inductive argument on the sequence of states).

If we follow a similar derivation to that given in equation (1) we arrive at the corresponding feature for document $d$ and transition on $x$ from $u$ of

$$\phi_{u,x}(d) = \frac{\mathrm{tf}((u,x),d)}{p_u(x)},$$

where we use $\mathrm{tf}((u,x),d)$ to denote the frequency of the transition on symbol $x$ from a state $u$ with non-unity $p_u$ in document $d$.

Hence, given an FSM model we can construct the corresponding Fisher kernel feature vector by simply processing the document through the FSM and recording the counts for each transition. The corresponding feature vector will be sparse relative to the dimension of the feature space (the total number of transitions in the FSM) since only those transitions actually used will have non-zero entries. Hence, as for the bag of words we can create feature vectors by listing the indices of transitions used followed by their frequency. The number of non-zero features will be at most equal to the number of symbols in the document.

Consider taking $\Sigma = \bigcup_{i=0}^{k-1} A^i$ with all the distributions $p_u$ uniform for $u \in A^{k-1}$ and $p_u \equiv 1$ for other $u$. In this case we recover the $k$-gram model and corresponding kernel.

A problem that we have observed when experimenting with the $n$-gram model is that if we estimate the frequencies of transitions from the corpus certain transitions can become very frequent while others from the same state occur only rarely. In such cases the rare states will receive a very high weighting in the Fisher score vector. One would like to use the strategy adopted for the `idf` weighting for the bag of words kernel which is often taken to be

$$\ln\left(\frac{m}{m_i}\right),$$

where $m$ is the number of documents and $m_i$ the number containing term $i$. The ln ensures that the contrast in weighting is controlled. We can obtain this effect in the Fisher kernel if we reparametrise the transition probabilities as follows

$$p_u(x) = \exp(-\exp(-t_u(x))),$$

where $t_u(x)$ is the new parameter. With this parametrisation the derivative of the ln probabilities becomes

$$\frac{\partial \ln p_u(x)}{\partial t_u(x)} = \exp(-t_u(x)) = -\ln p_u(x),$$

as required.

Although this improves performance the problem of frequent substrings being un-informative remains. We now consider the idea outlined above of moving to longer subsequences in order to ensure that transitions are informative.

## 4 Choosing Features

There is a critical frequency at which the most information is conveyed by a feature. If it is ubiquitous as we observed above it gives little or no information for analysing documents. If on the other hand it is very infrequent it again will not be useful since we are only rarely able to use it. The usefulness is maximal at the threshold between these two extremes. Hence, we would like to create states that occur not too frequently and not too infrequently.

A natural way to infer the set of such states is from the training corpus. We select all substrings that have occurred at least $t$ times in the document corpus, where $t$ is a small but statistically visible number. In our experiments we took $t = 10$.

Hence, given a corpus $\mathcal{S}$ we create the FSM model $\mathcal{F}_t(\mathcal{S})$ with

$$\Sigma_t(\mathcal{S}) = \{u \in A^\star : u \text{ occurs at least } t \text{ times in the corpus } \mathcal{S}\}.$$

Taking this definition of $\Sigma_t(\mathcal{S})$ we construct the corresponding finite state machine model as described in Definition 2. We will refer to the model $\mathcal{F}_t$ as the *frequent set FSM at threshold* $t$.

We now construct the transition probabilities by processing the corpus through the $\mathcal{F}_t(\mathcal{S})$ keeping a tally of the number of times each transition is actually used. Typically we initialise the counts to some constant value $c$ and convert the resulting counts into probabilities for the model. Hence, if $f_{u,x}$ is the number of times we leave state $u$ processing symbol $x$, the corresponding probabilities will be

$$p_u(x) = \frac{f_{u,x} + c}{|A|c + \sum_{x' \in A} f_{u,x'}}. \tag{2}$$

Note that we will usually exclude from the count the transitions at the beginning of a document $d$ that start from states $d[1:j]$ for some $j \geq 0$.

The following proposition demonstrates that the model has the desired frequency properties for the transitions. We use the notation $u \xrightarrow{x} v$ to indicate the transition from state $u$ to state $v$ on processing symbol $x$.

**Proposition 3** *Given a corpus $\mathcal{S}$ the FSM model $\mathcal{F}_t(\mathcal{S})$ satisfies the following property. Ignoring transitions from states indexed by $d[1:i]$ for some document $d$ of the corpus, the frequency counts $f_{u,x}$ for transitions $u \xrightarrow{x} v$ in the corpus $\mathcal{S}$ satisfy*

$$\sum_{x \in A} f_{u,x} < t|A|$$

*for all $u \in \Sigma_t(\mathcal{S})$.*

**Proof.** Suppose that for some state $u \in \Sigma_t(\mathcal{S})$

$$\sum_{x \in A} f_{u,x} \geq t|A| \tag{3}$$

This implies that the string $u$ has occurred at least $t|A|$ times at the head of a transition not at the beginning of a document. Hence, by the pigeon hole principle there is a $y \in A$ such that $y$ has occurred $t$ times immediately before one of the transitions in the sum of (3). Note that this also implies that $yu$ occurs at least $t$ times in the corpus and therefore will be in $\Sigma_t(\mathcal{S})$. Consider one of the transitions that occurs after $yu$ on some symbol $x$. This transition will not be of the form $u \xrightarrow{x} v$ but rather $yu \xrightarrow{x} v$ contradicting its inclusion in the sum (3). Hence, the proposition holds. ∎

Note that the proposition implies that no individual transition can be more frequent than the full sum. The proposition also has useful consequences for the maximum weighting for any Fisher score entries as the next corollary demonstrates.

**Corollary 4** *Given a corpus $\mathcal{S}$ if we construct the FSM model $\mathcal{F}_t(\mathcal{S})$ and compute the probabilities by counting transitions ignoring those from states indexed by $d[1:i]$ for some document $d$ of the corpus, the probabilities on the transitions will satisfy*

$$\min_{u,x} p_u(x) > \frac{c}{|A|(c+t)}.$$

**Proof.** We substitute the bound given in the proposition into the formula (2). ∎

The proposition and corollary demonstrate that the choice of $\mathcal{F}_t(\mathcal{S})$ as an FSM model has the desirable property that all of the states are meaningfully frequent while none of the transitions is too frequent and furthermore the Fisher weighting cannot grow too large for any individual transition.

In the next section we will present experimental results testing the kernels we have introduced using the standard and logarithmic weightings. The baseline for the experiments will always be the bag of words kernel using the TFIDF weighting scheme. It is perhaps worth noting that though the IDF weighting appears similar to those described above it makes critical use of the distribution of terms across documents, something that is incompatible with the Fisher approach that we have adopted. It is therefore very exciting to see the results that we are able to obtain using these syntactic features and sub-document level weightings.

## 5    Experimental Results

Our experiments were conducted on the top 10 categories of the standard Reuters-21578 data set using the "Mod Apte" split. We compared the standard n-gram kernel with a Uniform, non-uniform and ln weighting scheme, and the variable-length FSM model described in Section 4 both with uniform weighting and a ln weighting scheme. As mentioned in Section 4, the parameter $t$ was set to 10. In order to keep the comparison fair, the n-gram kernel features were also pruned from the feature vector if they occured less than 10 times. For our experiments we used 5-gram features, which have previously been reported to give the best results [5]. The standard bag of words model using the normal tfidf weighting scheme is used as a baseline. Once feature vectors had been created they were normalised and the SVMlight software package [3] was used with the default parameter settings to obtain outputs for the test examples. In order to compare algorithms, we used the average performance measure commonly used in Information Retrieval (see e.g. [4]). This is the average of precision values obtained when thresholding at each positively classified document. If all positive documents in the corpus are ranked higher than any negative documents, then the average precision is 100%. Average precision incorporates both precision and recall measures and is highly sensitive to document ranking, so therefore can be used to obtain a fair comparison between methods. The results are shown in Table 1.

As can bee seen from the table, the variable-length subsequence method performs as well as or better than all other methods and achieves a perfect ranking for documents in one of the categories.

| Method | BoW | ngrams | | | FSA | |
|---|---|---|---|---|---|---|
| Weighting | TFIDF | Uniform | $\frac{1}{p_i}$ | $\ln\frac{1}{p_i}$ | Uniform | $\ln\frac{1}{p_i}$ |
| earn | 99.86 | 99.91 | 96.4 | 99.9 | 99.9 | 99.9 |
| acq | 99.62 | 99.61 | 99.7 | 99.5 | 99.7 | 99.7 |
| money-fx | 80.54 | 82.43 | 84.9 | 83.4 | 86.5 | 85.8 |
| grain | 99.69 | 99.67 | 99.9 | 99.4 | 97.8 | 97.5 |
| crude | 98.52 | 98.23 | 99.9 | 97.2 | 100.0 | 100.0 |
| trade | 95.29 | 95.53 | 94.6 | 95.6 | 94.6 | 91.3 |
| interest | 91.61 | 98.83 | 96.6 | 95.4 | 94.0 | 88.8 |
| ship | 96.84 | 99.42 | 91.7 | 98.9 | 92.7 | 98.4 |
| wheat | 98.52 | 98.7 | 97.2 | 99.3 | 95.3 | 98.4 |
| corn | 98.95 | 98.2 | 99.3 | 99.0 | 97.5 | 98.1 |

Table 1: Average precision results comparing TFIDF, n-gram and FSM features on the top 10 categories of the reuters data set.

## 6 Discussion

In this paper we have shown how the string kernel can be thought of as a $k$-stage Markov process, and as a result interpreted as a Fisher kernel. Using this new insight we have shown how the features of a Fisher kernel can be constructed using a Finite State Model parameterisation which reflects the statistics of the frequency of occurance of features within the corpus. This model has then been extended further to incorporate sub-sequences of varying length, which is a great deal more flexible than the fixed-length approach. A procedure for determining informative sub-sequences (states in the FSM model) has also been given. Experimental results have shown that this model outperforms the standard tfidf bag of words model on a well known data set. Although the experiments in this paper are not extensive, they show that the approach of using a Finite-State-Model to generate a Fisher kernel gives new insights and more flexibility over the string kernel, and performs well. Future work would include determining the optimum value for the threshold $t$ (maximum frequency of a sub-string occurring within the FSM before a state is expanded) as this currently has to be set a-priori.

## References

[1] D. Haussler. Convolution kernels on discrete structures. Technical Report UCSC-CRL-99-10, University of California, Santa Cruz, July 1999.

[2] T. Jaakkola, M. Diekhaus, and D. Haussler. Using the fisher kernel method to detect remote protein homologies. *7th Intell. Sys. Mol. Biol.*, pages 149–158, 1999.

[3] T. Joachims. Making large-scale svm learning practical. In B. Schölkopf, C. Burges, and A. Smola, editors, *Advances in Kernel Methods - Support Vector Learning*. MIT-Press, 1999.

[4] Y. Li, H. Zaragoza, R. Herbrich, J. Shawe-Taylor, and J. Kandola. The perceptron algorithm with uneven margins. In *Proceedings of the Nineteenth International Conference on Machine Learning (ICML '02)*, 2002.

[5] H Lodhi, C. Saunders, J. Shawe-Taylor, N. Cristianini, and Watkins C. Text classification using string kernels. *Journal of Machine Learning Research*, (2):419–444, 2002.

[6] H. Lodhi, J. Shawe-Taylor, N. Cristianini, and C. Watkins. Text classification using string kernels. In T. K. Leen, T. G. Dietterich, and V. Tresp, editors, *Advances in Neural Information Processing Systems 13*, pages 563–569. MIT Press, 2001.

[7] C. Watkins. Dynamic alignment kernels. Technical Report CSD-TR-98-11, Royal Holloway, University of London, January 1999.
